# General Bounds on Bayes Errors for Regression with Gaussian Processes

**Manfred Opper**
Neural Computing Research Group
Dept. of Electronic Engineering
and Computer Science,
Aston University,
Birmingham, B4 7ET
United Kingdom
opperm@aston.ac.uk

**Francesco Vivarelli**
Centro Ricerche Ambientali
Montecatini,

via Ciro Menotti, 48
48023 Marina di Ravenna,
Italy
fvivarelli@cramont.it

## Abstract

Based on a simple convexity lemma, we develop bounds for different types of Bayesian prediction errors for regression with Gaussian processes. The basic bounds are formulated for a fixed training set. Simpler expressions are obtained for sampling from an input distribution which equals the weight function of the covariance kernel, yielding asymptotically tight results. The results are compared with numerical experiments.

## 1 Introduction

Nonparametric Bayesian models which are based on Gaussian priors on function spaces are becoming increasingly popular in the Neural Computation Community (see e.g.[2, 3, 4, 7, 1]). Since the model classes considered in this approach are infinite dimensional, the application of Vapnik - Chervonenkis type of methods to determine bounds for the learning curves is nontrivial and has not been performed so far (to our knowledge). In these methods, the target function to be learnt is fixed and input data are drawn independently at random from a fixed (unknown) distribution. The approach of this paper is different. Here, we assume that the *target* is actually drawn at random from a known prior distribution, and we are interested in developing simple bounds on the average prediction performance (with respect to the prior) which hold for a fixed set of inputs. Only at a later stage, an average over the input distribution is made.

## 2    Regression with Gaussian processes

To explain the Gaussian process scenario for regression problems [4], we assume that observations $y \in R$ at input points $x \in R^D$ are corrupted values of a function $\theta(x)$ by an independent Gaussian noise with variance $\sigma^2$. The appropriate stochastic model is given by the likelihood

$$p_\theta(y|x) = \frac{e^{-\frac{(y-\theta(x))^2}{2\sigma^2}}}{\sqrt{2\pi\sigma^2}}. \tag{1}$$

The goal of a learner is to give an estimate of the function $\theta(x)$, based on a set of observed example data $D_t = ((x_1, y_1), \ldots, (x_t, y_t))$. As the prior information about the unknown function $\theta(x)$ we asume that $\theta$ is a realization of a Gaussian random field with zero mean and covariance

$$C(x, x') = \mathbb{E}[\theta(x)\theta(x')]. \tag{2}$$

It is useful to expand the random functions as

$$\theta(x) = \sum_{k=0}^{\infty} w_k \phi_k(x) \tag{3}$$

in a complete set of deterministic functions $\phi_k(x)$ with random Gaussian coefficients $w_k$. As is well known, if the $\phi_k$ are chosen as orthonormal eigenfunctions of the integral equation

$$\int C(x, x')\phi_k(x')p(x')dx' = \lambda_k \phi_k(x), \tag{4}$$

with eigenvalues $\lambda_k$ and a nonnegative weight function $p(x)$, the a priori statistics of $w_l$ is simple. They are *independent* Gaussian variables which satisfy $\mathbb{E}[w_k w_l] = \lambda_k \delta_{kl}$.

## 3    Prediction and Bayes error

Usually, the posterior mean of $\theta(x)$ is chosen as the prediction $\hat{\theta}(x)$ on a new point $x$ based on a dataset $D_n = (x_1, y_1), \ldots, (x_n, y_n)$. Its explicit form can be easily derived by using the expansion $\hat{\theta}(x) = \sum_k \hat{w}_k \phi_k(x)$, and the fact that for Gaussian random variables, their mean coincides with their most probable value. Maximizing the log posterior, with respect to the $w_k$, one finds for the infinite dimensional vector $\hat{w} \doteq (w_k)_{k=0,\ldots,\infty}$ the result $\hat{w} = (\sigma^2 I + \Lambda V)^{-1} b$ where $V_{kl} = \sum_{i=1}^{n} \phi_k(x_i)\phi_l(x_i)$ $\Lambda_{kl} = \lambda_k \delta_{kl}$ and $b_k = \sum_{i=1}^{n} \lambda_k y_i \phi_k(x_i)$ Fixing the set of inputs $x^n$, the Bayesian prediction error at a point $x$ is given by

$$\varepsilon(x|x^n) \doteq \mathbb{E}\left(\theta(x) - \hat{\theta}(x)\right)^2 \tag{5}$$

Evaluating (5) yields, after some work, the expression

$$\varepsilon(x|x^n) = \sigma^2 \operatorname{Tr}\left\{\left(\sigma^2 I + \Lambda V\right)^{-1} \Lambda U(x)\right\} \tag{6}$$

with the matrix $U_{kl}(x) = \phi_k(x)\phi_l(x)$. $U$ has the properties that $\frac{1}{n}\sum_{i=1}^{n} U(x_i) = V$ and $\int dx\, p(x)U(x) = I$. We define the Bayesian *training error* as the empirical average of the error (5) at the $n$ datapoints of the training set and the Bayesian *generalization error* as the average error over all $x$ weighted by the function $p(x)$. We get

$$\varepsilon_t(x^n) = \frac{1}{n} \operatorname{Tr}\left\{\Lambda V \left(I + \Lambda V/\sigma^2\right)^{-1}\right\} \tag{7}$$

$$\varepsilon_g(x^n) = \operatorname{Tr}\left\{\Lambda \left(I + \Lambda V/\sigma^2\right)^{-1}\right\}. \tag{8}$$

## 4   Entropic error

In order to understand the next type of error [9], we assume that the data arrive sequentially, one after the other. The *predictive distribution* after $t-1$ training data at the new input $x_t$ is the posterior expectation of the likelihood (1), i.e.

$$\hat{P}(y|x_t, D_{t-1}) = \mathbb{E}[P_\theta(y|x_t)|D_{t-1}].$$

Let $L_t$ as the Bayesian average of the relative entropy (or Kullback Leibler divergence) between the predictive distribution and the true distribution $P_\theta$ from which the data were generated, i.e. $L_t = \mathbb{E}\left[D_{KL}\left(P_\theta||\hat{P}\right)\right]$. It can also be shown that $L_t = \frac{1}{2}\ln\left(1 + \frac{\varepsilon_g(x_t|x^{t-1})}{\sigma^2}\right)$. Hence, when the prediction error is small, we will have

$$L_t \approx \frac{1}{2}\frac{\varepsilon_g(x_t|x^{t-1})}{\sigma^2}. \tag{9}$$

The *cumulative* entropic error $E_e(x^n)$ is defined by summing up all the losses (which gives an integrated learning curve) from $t = 1$ up to time $n$ and one can show that

$$E(x_n) = \sum_{t=1}^{n} L_t(x_t, D_{t-1}) = \mathbb{E}D_{KL}\left(P_\theta^n||\hat{P}^n\right) = \frac{1}{2}\operatorname{Tr}\ln\left(I + \Lambda V/\sigma^2\right) \tag{10}$$

where $P_\theta^n = \prod_{i=1}^n P_\theta(y_i|x_i)$ and $\hat{P}^n = \mathbb{E}[\prod_{i=1}^n P_\theta(y_i|x_i)]$. The first equality may be found e.g. in [9], and the second follows from direct calculation.

## 5   Bounds for fixed set of inputs

In order to get bounds on (7),(8) and (10), we use a lemma, which has been used in Quantum Statistical Mechanics to get bounds on the free energy. The lemma (for the special function $f(x) = e^{-\beta x}$) was proved by Sir Rudolf Peierls in 1938 [10]. In order to keep the paper self contained, we have included the proof in the appendix.

**Lemma 1** *Let $H$ be a real symmetric matrix and $f$ a convex real function. Then* $\operatorname{Tr} f(H) \geq \sum_k f(H_{kk})$.

By noting, that for concave functions the bound goes in the other direction, we immediately get

$$\varepsilon_t \leq \frac{\sigma^2}{n}\sum_k \frac{\lambda_k V_{kk}}{\sigma^2 + \lambda_k V_{kk}} \leq \sigma^2 \sum_k \frac{\lambda_k v_k}{\sigma^2 + n\lambda_k v_k} \tag{11}$$

$$\varepsilon_g \geq \sum_k \frac{\sigma^2 \lambda_k}{\sigma^2 + \lambda_k V_{kk}} \geq \sum_k \frac{\sigma^2 \lambda_k}{\sigma^2 + n\lambda_k v_k} \tag{12}$$

$$E(x^n) \leq \frac{1}{2}\sum_k \ln\left(1 + V_{kk}\lambda_k/\sigma^2\right) \leq \frac{1}{2}\sum_k \ln\left(1 + nv_k\lambda_k/\sigma^2\right) \tag{13}$$

where in the rightmost inequalities, we assume that all $n$ inputs are in a compact region $\mathcal{D}$, and we define $v_k = \sup_{x\in\mathcal{D}} \phi_k^2(x)$. [1]

# 6   Average case bounds

Next, we assume that the input data are drawn at random and denote by $\langle \ldots \rangle$ the expectations with respect to the distribution. *We do not have to assume independence here, but only the fact that all marginal distributions for the $n$ inputs are identical!* Using Jensen's inequality

$$\varepsilon_t = \langle \varepsilon_t(x^n) \rangle \leq \sigma^2 \sum_k \frac{\lambda_k u_k}{\sigma^2 + n\lambda_k u_k} \tag{14}$$

$$\varepsilon_g = \langle \varepsilon_g(x^n) \rangle \geq \sum_k \frac{\sigma^2 \lambda_k}{\sigma^2 + n\lambda_k u_k} \tag{15}$$

$$E = \langle E(x^n) \rangle \leq \frac{1}{2} \sum_k \ln \left(1 + n u_k \lambda_k / \sigma^2\right) \tag{16}$$

where now $u_k = \langle \phi_k^2(x) \rangle$. This result is especially simple, when the weighting function $p(x)$ is a probability density and the inputs have the marginal distribution $p(x)$. In this case, we simply have $u_k = 1$. In this case, training and generalization error sandwich the bound

$$\varepsilon_b = \sigma^2 \sum_k \frac{\lambda_k}{\sigma^2 + n\lambda_k}. \tag{17}$$

We expect that the bound $\varepsilon_b$ becomes asymptotically exact, when $n \to \infty$. This should be intuitively clear, because training and generalization error approach each other asymptotically. This fact may also be understood from (9), which shows that the cumulative entropic error is within a factor of $\frac{1}{2}$ asymptotically equal to the cumulative generalization error. By integrating the lower bound (17) over $n$, we obtain precisely the upper bound on $E$ with a factor 2, showing that upper and lower bounds show the same behaviour.

# 7   Simulations

We have compared our bounds with simulations for the average training error and generalization error for the case that the data are drawn from $p(x)$. Results for the entropic error will be given elsewhere.

We have specialized on the case, where the covariance kernel is of the RBF form $C(x, x') = \exp[(x - x')^2 / \lambda^2]$, and $p(x) = (2\pi)^{-\frac{1}{2}} e^{-\frac{1}{2}x^2}$, for which, following Zhu *et al.* (1997), the $k$-th eigenvalue of the spectrum ($k = 0 \ldots \infty$) can be written as $\lambda_k = ab^k$, where $a = \sqrt{c}, b = c/\lambda^2$, $c = 2 \left(1 + 2/\lambda^2 + \sqrt{1 + 4/\lambda^2}\right)^{-1}$, and $\lambda$ is the lengthscale of the process. We estimated the average generalisation error for each training set based on the exact analytical expressions (8) and (7) over the distribution of the datasets by using a Monte Carlo approximation. To begin with, let us consider $x \in R$. We sampled the 1−dimensional input space generating 100 training sets whose data points were normally distributed around zero with unit variance. For each generation, the expected training and generalisation errors for a GP have been evaluated using up to 1000 data points. We set the value of the lengthscale² $\lambda$ to 0.1 and we let the noise level $\sigma^2$ assume several values ($\sigma^2 = 10^{-4}, 10^{-3}, 10^{-2}, 10^{-1}, 1$). Figure 1 shows the results we obtained when

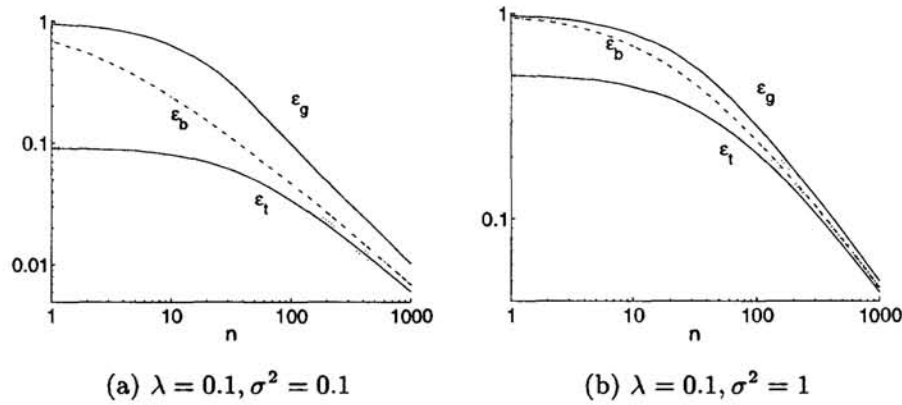

(a) $\lambda = 0.1, \sigma^2 = 0.1$                    (b) $\lambda = 0.1, \sigma^2 = 1$

Figure 1: The Figures show the graphs of the training and learning curves with their bound $\epsilon_b(n)$ obtained with $\lambda = 0.1$; the noise level is set to 0.1 in Figure 1(a) and to 1 in Figure 1(b). In all the graphs, $\epsilon_t$ and $\epsilon_g(n)$ are drawn by the solid line and their 95% confidence interval is signed by the dotted curves. The bound $\epsilon_b(n)$ is drawn by the dash-dotted lines.

$\sigma^2 = 0.1$ (Figure 1(a)) and $\sigma^2 = 1$ (Figure 1(b)). The bound $\epsilon_b(n)$ lies within the training and learning curves, being an upper bound for $\epsilon_t(n)$ and a lower bound for $\epsilon_g(n)$. This bound is tighter for the processes with higher noise level; in particular, for large datasets the error bars on the curves $\epsilon_t(n)$ and $\epsilon_g(n)$ overlap the bound $\epsilon_b(n)$. The curves $\epsilon_t(n)$, $\epsilon_g(n)$ and $\epsilon_b(n)$ approach zero as $O(\log(n)/n)$.

Our bounds can also be applied to higher dimensions $D > 1$ using the covariance

$$C(x, x') = \exp\left(-\|x - x'\|^2/\lambda^2\right) \tag{18}$$

for $x, x' \in R^D$. Obviously the integral kernel $C$ is just a direct product of RBF kernels, one for each coordinate of $x$ and $x'$. The eigenvalue problem (4) can be immediately reduced to the one for a single variable. Eigenfunctions and eigenvalues are simply products of those for the single coordinate problems. Hence, using a bit of combinatorics, the bound $\varepsilon_b$ can be written as

$$\varepsilon_b = \sum_{k=0}^{\infty} \binom{k + D - 1}{k} \frac{\sigma^2 a^D b^k}{\sigma^2 + n a^D b^k}, \tag{19}$$

where $a$ and $b$ have been defined above. We performed experiments when $x \in R^2$ and $x \in R^5$. The correlation lengths along each direction of the input space has been set to 1 and the noise level was $\sigma^2 = 1.0$. The graphs of the curves, with their error bars are reported in Figure 2(a) (for $x \in R^2$) and in Figure 2(b) (for $x \in R^5$).

## 8   Discussion

Based on the minimal requirements on training inputs and covariances, we conjecture that our bounds cannot be improved much without making more detailed assumptions on models and distributions. We can observe from the simulations that the tightness of the bound $\epsilon_b(n)$ depends on the dimension of the input space. In particular, for large datasets $\epsilon_b(n)$ is tighter for small dimension of the input space; Figure 2(a) shows this quite clearly since $\epsilon_b(n)$ overlaps the error bars of the

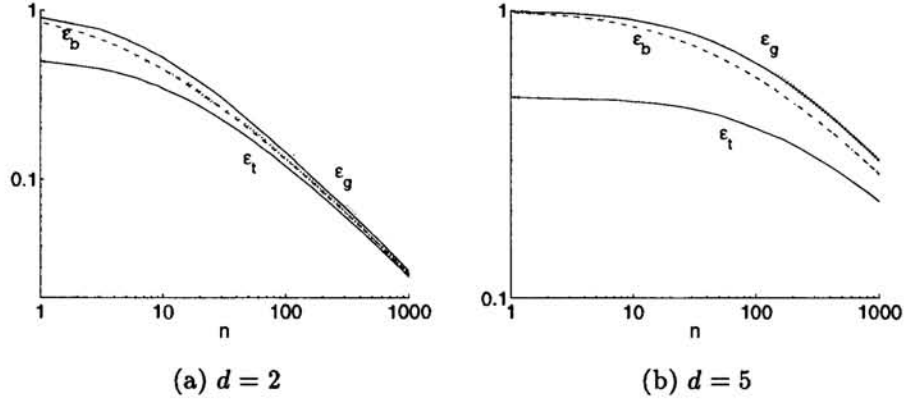

(a) $d = 2$     (b) $d = 5$

Figure 2: The Figures show the graphs of the training and learning curves with their bound $\epsilon_b(n)$ obtained with the squared exponential covariance function with $\lambda = 1$ and $\sigma^2 = 1$; the input space is $R^2$ (Figure 2(a)) and $R^5$ (Figure 2(b)). In all the Figures, $\epsilon_t$ and $\epsilon_g(n)$ are drawn by the solid line and their 95% confidence interval is signed by the dotted curves. The bound $\epsilon_b(n)$ is drawn by the dash-dotted lines.

training and learning curves for large $n$. Numerical simulations performed using modified Bessel covariance functions of order $r$ (describing random processes $r - 1$ time mean square differentiable) have shown that the bound $\epsilon_b(n)$ becomes tighter for smoother processes.

**Acknowledgement:** We are grateful for many inspiring discussions with C.K.I. Williams. M.O. would like to thank Peter Sollich for his conjecture that (17) is an exact lower bound on the generalization error, which motivated part of this work. F. V. was supported by a studentship of British Aerospace.

## 9   Appendix: Proof of the lemma 1

Let $\{\Phi^{(j)}\}$ be a complete set of orthonormal eigenvectors and $\{E_i\}$ the corresponding set of eigenvalues of $H$, i.e. we have the properties $\sum_l H_{kl} \Phi_l^{(i)} = E_i \Phi_k^{(i)}$, $\sum_i \Phi_k^{(i)} \Phi_l^{(i)} = \delta_{kl}$, and $\sum_k \Phi_k^{(i)} \Phi_k^{(j)} = \delta_{ij}$. Then we get

$$
\begin{aligned}
\text{Tr } f(H) &= \sum_i f(E_i) = \sum_k \sum_i (\Phi_k^{(i)})^2 f(E_i) \\
&\geq \sum_k f\left(\sum_i (\Phi_k^{(i)})^2 E_i\right) = \sum_k f\left(\sum_i \Phi_k^{(i)} \sum_l H_{kl} \Phi_l^{(i)}\right) \\
&= \sum_k f(H_{kk})
\end{aligned}
$$

The second equality follows from orthonormality, because $\sum_k (\Phi_k^{(i)})^2 = 1$. The inequality uses the fact that by completeness, for any $k$, we have $\sum_i (\Phi_k^{(i)})^2 = 1$ and we may regard the $(\Phi_k^{(i)})^2$ as probabilities, such that by convexity, Jensen's inequality can be used. After using the eigenvalue equation, the sum over $i$ was carried out with the help of the completeness relation, in order to obtain the last line.

## Footnotes

[1]The entropic case may also be proved by Hadamard's inequality.

[2] The value of the lengthscale $\lambda$ has the effect of stretching the training and learning curves; thus the results of the experiments performed with different $\lambda$ are qualitatively similar to those presented.

# References

[1] D. J. C. Mackay, Gaussian Processes, A Replacement for Neural Networks, NIPS tutorial 1997. May be obtained from `http://wol.ra.phy.cam.ac.uk/pub/mackay/`.

[2] R. Neal, *Bayesian Learning for Neural Networks*, Lecture Notes in Statistics, Springer (1996).

[3] C. K. I. Williams, Computing with Infinite Networks, in *Neural Information Processing Systems 9*, M. C. Mozer, M. I. Jordan and T. Petsche, eds., 295-301. MIT Press (1997).

[4] C. K. I. Williams and C. E. Rasmussen, Gaussian Processes for Regression, in *Neural Information Processing Systems 8*, D. S. Touretzky, M. C. Mozer and M. E. Hasselmo eds., 514-520, MIT Press (1996).

[5] R. M. Neal, Monte Carlo Implementation of Gaussian Process Models for Bayesian Regression and Classification, Technical Report CRG-TR-97-2, Dept. of Computer Science, University of Toronto (1997).

[6] M. N. Gibbs and D. J. C. Mackay, Variational Gaussian Process Classifiers, Preprint Cambridge University (1997).

[7] D. Barber and C. K. I. Williams, Gaussian Processes for Bayesian Classification via Hybrid Monte Carlo, in *Neural Information Processing Systems 9*, M. C. Mozer, M. I. Jordan and T. Petsche, eds., 340-346. MIT Press (1997).

[8] C. K. I. Williams and D. Barber, Bayesian Classification with Gaussian Processes, Preprint Aston University (1997).

[9] D. Haussler and M. Opper, Mutual Information, Metric Entropy and Cumulative Relative Entropy Risk, The Annals of Statistics, Vol 25, No 6, 2451 (1997).

[10] R. Peierls, Phys. Rev. 54, 918 (1938).

[11] H. Zhu, C. K. I. Williams, R. Rohwer and M. Morciniec, Gaussian Regression and Optimal Finite Dimensional Linear Models, Technical report NCRG/97/011, Aston University (1997).
